# Regularized Co-Clustering with Dual Supervision

**Vikas Sindhwani      Jianying Hu      Aleksandra Mojsilovic**
IBM Research, Yorktown Heights, NY 10598
{vsindhw, jyhu, aleksand}@us.ibm.com

## Abstract

By attempting to simultaneously partition both the rows (examples) and columns (features) of a data matrix, Co-clustering algorithms often demonstrate surprisingly impressive performance improvements over traditional one-sided row clustering techniques. A good clustering of features may be seen as a combinatorial transformation of the data matrix, effectively enforcing a form of regularization that may lead to a better clustering of examples (and vice-versa). In many applications, partial supervision in the form of a few row labels as well as column labels may be available to potentially assist co-clustering. In this paper, we develop two novel semi-supervised multi-class classification algorithms motivated respectively by spectral bipartite graph partitioning and matrix approximation formulations for co-clustering. These algorithms (i) support dual supervision in the form of labels for both examples and/or features, (ii) provide principled predictive capability on out-of-sample test data, and (iii) arise naturally from the classical Representer theorem applied to regularization problems posed on a collection of Reproducing Kernel Hilbert Spaces. Empirical results demonstrate the effectiveness and utility of our algorithms.

## 1   Introduction

Consider the setting where we are given large amounts of unlabeled data together with dual supervision in the form of a few labeled examples *as well as a few labeled features*, and the goal is to estimate an unknown classification function. This setting arises naturally in numerous applications. Imagine, for example, the problem of inferring sentiment ("positive" versus "negative") associated with presidential candidates from online political blog posts represented as word vectors, given the following: (a) a vast collection of blog posts easily downloadable from the web (unlabeled examples), (b) a few blog posts whose sentiment for a candidate is manually identified (labeled examples), and (c) prior knowledge of words that reflect positive (e.g., *'superb'*) and negative (e.g, *'awful'*) sentiment (labeled features). Most existing semi-supervised algorithms do not explicitly incorporate feature supervision. They typically implement the *cluster assumption* [3] by learning decision boundaries such that unlabeled points belonging to the same cluster are given the same label, and empirical loss over labeled examples is concurrently minimized. In situations where the classes are predominantly supported on unknown subsets of similar features, it is clear that feature supervision can potentially illuminate the true cluster structure inherent in the unlabeled examples over which the cluster assumption ought to be enforced.

Even when feature supervision is not available, there is ample empirical evidence in numerous recent papers in the co-clustering literature (see e.g.,  [5, 1] and references therein), suggesting that the clustering of columns (features) of a data matrix can lead to massive improvements in the quality of row (examples) clustering. An intuitive explanation is that column clustering enforces a form of dimensional reduction or implicit regularization that is responsible for performance enhancements observed in many applications such as text clustering, microarray data analysis and video content mining [1]. In this paper, we utilize data-dependent co-clustering regularizers for semi-supervised learning in the presence of partial dual supervision.

Our starting point is the spectral bipartite graph partitioning approach of [5] which we briefly review in Section 2.1. This approach effectively applies spectral clustering on a graph representation of the data matrix and is also intimately related to Singular Value Decomposition. In Section 2.2 we review an equivalence between this approach and a matrix approximation objective function that is minimized under orthogonality constraints [6]. By dropping the orthogonality constraints but imposing non-negativity constraints, one is led to a large family of co-clustering algorithms that arise from the non-negative matrix factorization literature.

Based on the algorithmic intuitions embodied in the algorithms above, we develop two semi-supervised classification algorithms that extend the spectral bipartite graph partitioning approach and the matrix approximation approach respectively. We start with Reproducing Kernel Hilbert Spaces (RKHSs) defined *over both row and column spaces*. These RKHSs are then coupled through co-clustering regularizers. In the first algorithm, we directly adopt graph Laplacian regularizers constructed from the bipartite graph of [5] and include it as a row and column smoothing term in the standard regularization objective function. The solution is obtained by solving a convex optimization problem. This approach may be viewed as a modification of the Manifold Regularization framework [2] where we now jointly learn row and column classification functions. In the second algorithm proposed in this paper, we instead add a (non-convex) matrix approximation term to the objective function, which is then minimized using a block-coordinate descent procedure.

Unlike, their unsupervised counterparts, our methods support dual supervison and naturally possess out-of-sample extension. In Section 4, we provide experimental results where we compare against various baseline approaches, and highlight the performance benefits of feature supervision.

## 2 Co-Clustering Algorithms

Let $\mathbf{X}$ denote the data matrix with $n$ data points and $d$ features. The methods that we discuss in this section output a row partition function $\pi_r : \{i\}_{i=1}^n \mapsto \{j\}_{j=1}^{m_r}$ and a column partition function $\pi_c : \{i\}_{i=1}^d \mapsto \{j\}_{j=1}^{m_c}$ that give cluster assignments to row and column indices respectively. Here, $m_r$ is the desired number of row clusters and $m_c$ is the desired number of column clusters. Below, by $\boldsymbol{x}_i$ we mean the $i^{th}$ example (row) and by $\boldsymbol{f}_j$ we mean the $j^{th}$ column (feature) in the data matrix.

### 2.1 Bipartite Graph Partitioning

In the co-clustering technique introduced by [5], the data matrix is modeled as a bipartite graph with examples (rows) as one set of nodes and features (columns) as another. An edge $(i, j)$ exists if feature $\boldsymbol{f}_j$ assumes a non-zero value in example $\boldsymbol{x}_i$, in which case the edge is given a weight of $\mathbf{X}_{ij}$. This bi-partite graph is undirected and there are no inter-example or inter-feature edges. The adjacency matrix, $\mathbf{W}$, and the normalized Laplacian [4], $\mathbf{M}$, of this graph are given by,

$$\mathbf{W} = \begin{bmatrix} \mathbf{0} & \mathbf{X} \\ \mathbf{X}^T & \mathbf{0} \end{bmatrix}, \quad \mathbf{M} = \mathbf{I} - \mathbf{D}^{-\frac{1}{2}} \mathbf{W} \mathbf{D}^{-\frac{1}{2}} \tag{1}$$

where $\mathbf{D}$ is the diagonal degree matrix defined by $\mathbf{D}_{ii} = \sum_i \mathbf{W}_{ij}$ and $\mathbf{I}$ is the $(n+d) \times (n+d)$ identity matrix. Guided by the premise that column clustering induces row clustering while row clustering induces column clustering, [5] propose to find an optimal partitioning of the nodes of the bipartite graph. This method is retricted to obtaining co-clusterings where $m_r = m_c = m$. The $m$-partitioning is obtained by minimizing the relaxation of the normalized cut objective function using standard spectral clustering techniques. This reduces to first constructing a spectral representation of rows and columns given by the smallest eigenvectors of $\mathbf{M}$, and then performing standard $k$-means clustering on this representation, to finally obtain the partition functions $\pi_r, \pi_c$. Due to the special structure of Eqn. 1, it can be shown that the spectral representation used in this algorithm is related to the singular vectors of a normalized version of $\mathbf{X}$.

### 2.2 Matrix Approximation Formulation

In [6] it is shown that the bipartite spectral graph partitioning is closely related to solving the following matrix approximation problem, $(\mathbf{F}_r{}^\star, \mathbf{F}_c{}^\star) = \operatorname{argmin}_{\mathbf{F}_r{}^T \mathbf{F}_r = \mathbf{I}, \mathbf{F}_c{}^T \mathbf{F}_c = I} \|\mathbf{X} - \mathbf{F}_r \mathbf{F}_c{}^T\|_{fro}$ where $\mathbf{F}_r$ is an $n \times m$ matrix and $\mathbf{F}_c$ is a $d \times m$ matrix. Once the minimization is performed,

$\pi_r(i) = \text{argmax}_j \mathbf{F}_{r_{ij}}^\star$ and $\pi_c(i) = \text{argmax}_j \mathbf{F}_{c_{ij}}^\star$. In a non-negative matrix factorization approach, the orthogonality constraints are dropped to make the optimization easier while non-negativity constraints $\mathbf{F}_r, \mathbf{F}_c \geq 0$ are introduced with the goal of lending better interpretability to the solutions. There are numerous multiplicative update algorithms for NMF which essentially have the flavor of alternating non-convex optimization. In our empirical comparisons in Section 4, we use the Alternating Constrained Least Squares (ACLS) approach of [12]. In Section 3.2 we consider a 3-factor non-negative matrix approximation to incorporate unequal values of $m_r$ and $m_c$, and to improve the quality of the approximation. See [7, 13] for more details on matrix tri-factorization based formulations for co-clustering.

## 3  Objective Functions for Regularized Co-clustering with Dual Supervision

Let us consider examples $\boldsymbol{x}$ to be elements of $\mathcal{R} \subset \Re^d$. We consider column values $\boldsymbol{f}$ for each feature to be a data point in $\mathcal{C} \subset \Re^n$. Our goal is to learn partition functions defined over the entire row and column spaces (as opposed to matrix indices), i.e., $\pi_r : \mathcal{R} \mapsto \{i\}_{i=1}^{m_r}$ and $\pi_c : \mathcal{C} \mapsto \{i\}_{i=1}^{m_c}$. For this purpose, let us introduce $k_r : \mathcal{R} \times \mathcal{R} \to \Re$ to be the row kernel that defines an associated RKHS $\mathcal{H}_r$. Similarly, $k_c : \mathcal{C} \times \mathcal{C} \to \Re$ denotes the column kernel whose associated RKHS is $\mathcal{H}_c$. Below, we define $\pi_r, \pi_c$ using these real valued function spaces.

Consider a simultaneous assignment of rows into $m_r$ classes and columns into $m_c$ classes. For any data point $\boldsymbol{x}$, denote $F_r(\boldsymbol{x}) = [f_r^1(\boldsymbol{x}) \cdots f_r^{m_r}(\boldsymbol{x})]^T \in \Re^{m_r}$ to be a vector whose elements are soft class assignments where $f_r^j \in \mathcal{H}_r$ for all $j$. For the given $n$ data points, denote $\mathbf{F}_r$ to be the $n \times m_r$ class assignment matrix. Correspondingly, $F_c(\boldsymbol{f})$ is defined for any feature $\boldsymbol{f} \in \mathcal{C}$, and $\mathbf{F}_c$ denotes the associated column class assignment matrix. Additionally, we are given dual supervision in the form of label matrices $\mathbf{Y}_r \in \Re^{n \times m_r}$ and $\mathbf{Y}_c \in \Re^{m \times m_c}$ where $\mathbf{Y}_{r_{ij}} = 1$ specifies that the $i^{th}$ example is labeled with class $j$ (similarly for the feature labels matrix $\mathbf{Y}_c$). The associated row sum for a labeled point is 1. Unlabeled points have all-zero rows, and the row sums are therefore 0. Let $J_r$ ($J_c$) denote a diagonal matrix of size $n \times n$ ($d \times d$) whose diagonal entry is 1 for labeled examples (features) and 0 otherwise. By $I_s$ we will denote an identity matrix of size $s \times s$. We use the notation $\text{tr}(A)$ to mean the trace of the matrix $A$.

### 3.1  Manifold Regularization with Bipartite Graph Laplacian (MR)

In this approach, we setup the following optimization problem,

$$\operatorname*{argmin}_{F_r \in \mathcal{H}_r^{m_r}, F_c \in \mathcal{H}_c^{m_c}} \frac{\gamma_r}{2} \sum_{i=1}^{m_r} \|f_r^i\|_{\mathcal{H}_r}^2 + \frac{\gamma_c}{2} \sum_{i=1}^{m_c} \|f_c^i\|_{\mathcal{H}_c}^2 + \frac{1}{2}\text{tr}\left[(\mathbf{F}_r - \mathbf{Y}_r)^T J_r (\mathbf{F}_r - \mathbf{Y}_r)\right]$$
$$+ \frac{1}{2}\text{tr}\left[(\mathbf{F}_c - \mathbf{Y}_c)^T J_c (\mathbf{F}_c - \mathbf{Y}_c)\right] + \frac{\mu}{2}\text{tr}\left[\left(\mathbf{F}_r^T \mathbf{F}_c^T\right) \mathbf{M} \begin{pmatrix} \mathbf{F}_r \\ \mathbf{F}_c \end{pmatrix}\right] \qquad (2)$$

The first two terms impose the usual RKHS norm on the class indicator functions for rows and columns. The middle two terms measure squared loss on labeled data. The final term measure smoothness of the row and column indicator functions with respect to the bipartite graph introduced in Section 2.1. This term also incorporates unlabeled examples and features. $\gamma_r, \gamma_c, \mu$ are real-valued parameters that tradeoff various regularization terms.

Clearly, by Representer Theorem the solution is has the form,

$$f_r^j(\boldsymbol{x}) = \sum_{i=1}^{n} \alpha_{ij} k_r(\boldsymbol{x}, \boldsymbol{x}_i), 1 \leq j \leq m_r, \quad f_c^j(\boldsymbol{f}) = \sum_{i=1}^{d} \beta_{ij} k_c(\boldsymbol{f}, \boldsymbol{f}_i), 1 \leq j \leq m_c \qquad (3)$$

Let $\boldsymbol{\alpha}, \boldsymbol{\beta}$ denote the corresponding optimal expansion coefficient matrices. Then, plugging in Eqn. 3 and solving the optimization problem, the solution is easily seen to be given by,

$$\left[\begin{pmatrix} \gamma_r I_n & \mathbf{0} \\ \mathbf{0} & \gamma_c I_d \end{pmatrix} + \mu \mathbf{M} \begin{pmatrix} \mathbf{K}_r & \mathbf{0} \\ \mathbf{0} & \mathbf{K}_c \end{pmatrix} + \begin{pmatrix} J_r \mathbf{K}_r & \mathbf{0} \\ \mathbf{0} & J_c \mathbf{K}_c \end{pmatrix}\right] \begin{pmatrix} \boldsymbol{\alpha} \\ \boldsymbol{\beta} \end{pmatrix} = \begin{pmatrix} \mathbf{Y}_r \\ \mathbf{Y}_c \end{pmatrix} \qquad (4)$$

where $\mathbf{K}_r, \mathbf{K}_c$ are gram matrices over datapoints and features respectively. The partition functions are then defined by

$$\pi_r(\boldsymbol{x}) = \underset{1 \leq j \leq m}{\operatorname{argmax}} \sum_{i=1}^{n} \alpha_{ij} k_r(\boldsymbol{x}, \boldsymbol{x}_i), \qquad \pi_c(\boldsymbol{f}) = \underset{1 \leq j \leq m}{\operatorname{argmax}} \sum_{i=1}^{d} \beta_{ij} k_c(\boldsymbol{f}, \boldsymbol{f}_i) \qquad (5)$$

As in Section 2.1, we assume $m_r = m_c = m$. If the linear system above is solved by explicitly computing the matrix inverse, the computational cost is $O((n+d)^3 + (n+d)^2 m)$. This approach is closely related to the Manifold Regularization framework of [2], and may be viewed as an modification of the Laplacian Regularized Least Squares (LAPRLS) algorithm, which uses a euclidean nearest neighbor *row similarity* graph to capture the manifold structure in the data. Instead of using the squared loss, one can develop variants using the SVM Hinge loss or the logistic loss function. One can also use a large family of graph regularizers derived from the graph Laplacian [3]. In particular, we use the iterated Laplacian of the form $M^p$ where $p$ is an integer.

### 3.2 Matrix Approximation under Dual Supervision (MA)

We now consider an alternative objective function where instead of the graph Laplacian regularizer, we add a penalty term that measures how well the data matrix is approximated by a trifactorization $\mathbf{F}_r \mathbf{Q} \mathbf{F}_c^{\ T}$,

$$\underset{\substack{F_r \in \mathcal{H}_r^{m_r}, F_c \in \mathcal{H}_c^{m_c} \\ \mathbf{Q} \in \Re^{m_r \times m_c}}}{\operatorname{argmin}} \frac{\gamma_r}{2} \sum_{i=1}^{m_r} \|f_r^i\|_{\mathcal{H}_r}^2 + \frac{\gamma_c}{2} \sum_{i=1}^{m_c} \|f_c^i\|_{\mathcal{H}_c}^2 + \frac{1}{2} \operatorname{tr} \left[ (\mathbf{F}_r - \mathbf{Y}_r)^T J_r (\mathbf{F}_r - \mathbf{Y}_r) \right]$$

$$+ \frac{1}{2} \operatorname{tr} \left[ (\mathbf{F}_c - \mathbf{Y}_c)^T J_c (\mathbf{F}_c - \mathbf{Y}_c) \right] + \frac{\mu}{2} \|\mathbf{X} - \mathbf{F}_r \mathbf{Q} \mathbf{F}_c^{\ T}\|_{fro}^2 \qquad (6)$$

As before, the first two terms above enforce smoothness, the third and fourth terms measure squared loss over labels and the final term enforces co-clustering. The classical Representer Theorem (Eqn. 3) can again be applied since the above objective function only depends on point evaluations and RKHS norms of functions in $\mathcal{H}_r, \mathcal{H}_c$. The optimal expansion coefficient matrices, $\boldsymbol{\alpha}, \boldsymbol{\beta}$, in this case are obtained by solving,

$$\underset{\boldsymbol{\alpha}, \boldsymbol{\beta}, \mathbf{Q}}{\operatorname{argmin}} \mathcal{J}(\boldsymbol{\alpha}, \boldsymbol{\beta}, \mathbf{Q}) = \frac{\gamma_r}{2} \operatorname{tr} \left[ \boldsymbol{\alpha}^T \mathbf{K}_r \boldsymbol{\alpha} \right] + \frac{\gamma_c}{2} \operatorname{tr} \left[ \boldsymbol{\beta}^T \mathbf{K}_c \boldsymbol{\beta} \right] + \frac{1}{2} \operatorname{tr} \left[ (\mathbf{K}_r \boldsymbol{\alpha} - \mathbf{Y}_r)^T J_r (\mathbf{K}_r \boldsymbol{\alpha} - \mathbf{Y}_r) \right]$$

$$+ \frac{1}{2} \operatorname{tr} \left[ (\mathbf{K}_c \boldsymbol{\beta} - \mathbf{Y}_c)^T J_c (\mathbf{K}_c \boldsymbol{\beta} - \mathbf{Y}_c) \right] + \frac{\mu}{2} \|\mathbf{X} - \mathbf{K}_r \boldsymbol{\alpha} \mathbf{Q} \boldsymbol{\beta}^T \mathbf{K}_c\|_{fro}^2 \qquad (7)$$

This problem is not convex in $\boldsymbol{\alpha}, \boldsymbol{\beta}, \mathbf{Q}$. We propose a block coordinate descent algorithm for the problem above. Keeping two variables fixed, the optimization over the other is a convex problem with a unique solution. This guarantees monotonic decrease of the objective function and convergence to a stationary point. We get the simple update equations given below,

$$\frac{\partial \mathcal{J}}{\partial \mathbf{Q}} = 0 \quad \Longrightarrow \quad \mathbf{Q} = (\boldsymbol{\alpha}^T \mathbf{K}_r^2 \boldsymbol{\alpha})^{-1} (\boldsymbol{\alpha}^T \mathbf{K}_r \mathbf{X} \mathbf{K}_c \boldsymbol{\beta}) (\boldsymbol{\beta}^T \mathbf{K}_c^2 \boldsymbol{\beta})^{-1} \qquad (8)$$

$$\frac{\partial \mathcal{J}}{\partial \boldsymbol{\alpha}} = 0 \quad \Longrightarrow \quad [\gamma_r I_n + J_r \mathbf{K}_r] \, \boldsymbol{\alpha} + \mu \mathbf{K}_r \boldsymbol{\alpha} \mathbf{Z}_c = (J_r \mathbf{Y}_r + \mu \mathbf{X} \mathbf{K}_c \boldsymbol{\beta} \mathbf{Q}^T) \qquad (9)$$

$$\frac{\partial \mathcal{J}}{\partial \boldsymbol{\beta}} = 0 \quad \Longrightarrow \quad [\gamma_c I_d + J_c \mathbf{K}_c] \, \boldsymbol{\beta} + \mu \mathbf{K}_c \boldsymbol{\beta} \mathbf{Z}_r = (J_c \mathbf{Y}_c + \mu \mathbf{X}^T \mathbf{K}_r \boldsymbol{\alpha} \mathbf{Q}) \qquad (10)$$

$$\text{where } \mathbf{Z}_c = \mathbf{Q} \boldsymbol{\beta}^T \mathbf{K}_c^2 \boldsymbol{\beta} \mathbf{Q}^T, \ \mathbf{Z}_r = \mathbf{Q}^T \boldsymbol{\alpha}^T \mathbf{K}_r^2 \boldsymbol{\alpha} \mathbf{Q} \qquad (11)$$

In Eqn 8, we assume that the appropriate matrix inverses exist. Eqns 9 and 10 are generalized Sylvester matrix equations of the form $AXB^\top + CXD^\top = E$ whose unique solution $X$ under certain regularity conditions can be exactly obtained by an extended version of the classical Bartels-Stewart method [9] whose complexity is $O((p+q)^3)$ for $p \times q$-sized matrix variable $X$. Alternatively, one can solve the linear system [10]: $(B^\top \otimes A + D^\top \otimes C) \operatorname{vec}(X) = \operatorname{vec}(E)$ where $\otimes$ denotes

Kronecker product and vec($X$) vectorizes $X$ in a column oriented way (it behaves as the matlab operator $X(:)$). Thus, the solution to Eqns (9,10) are as follows,

$$[I_{m_r} \otimes (\gamma_r I_n + J_r \mathbf{K}_r) + \mu \mathbf{Z}_c \otimes \mathbf{K}_r] \, \text{vec}(\boldsymbol{\alpha}) \quad = \quad \text{vec}(J_r \mathbf{Y}_r + \mu \mathbf{X} \mathbf{K}_c \boldsymbol{\beta} \mathbf{Q}^T) \qquad (12)$$

$$[I_{m_c} \otimes (\gamma_r I_d + J_c \mathbf{K}_c) + \mu \mathbf{Z}_r \otimes \mathbf{K}_c] \, \text{vec}(\boldsymbol{\beta}) \quad = \quad \text{vec}(J_c \mathbf{Y}_c + \mu \mathbf{X}^T \mathbf{K}_r \boldsymbol{\alpha} \mathbf{Q}) \qquad (13)$$

These linear systems are of size $nm_r \times nm_r$ and $dm_c \times dm_c$ respectively. It is computationally prohibitive to solve these systems by direct matrix inversion. We use an iterative conjugate gradients (CG) technique instead, which can exploit hot-starts from the previous solution, and the fact that the matrix vector products can be computed relatively efficiently as follows,

$$[I_{m_r} \otimes (\gamma_r I_n + J_r \mathbf{K}_r) + \mu \mathbf{Z}_c \otimes \mathbf{K}_r] \, \text{vec}(\boldsymbol{\alpha}) = \text{vec}(\mu \mathbf{K}_r \boldsymbol{\alpha} \mathbf{Z}_c^\top) + \gamma_r \text{vec}(\boldsymbol{\alpha}) + \text{vec}(J_r \mathbf{K}_r \boldsymbol{\alpha})$$

To optimize $\boldsymbol{\alpha}$ ($\boldsymbol{\beta}$) given fixed $\mathbf{Q}$ and $\boldsymbol{\beta}$ ($\boldsymbol{\alpha}$), we run CG with a stringent tolerance of $10^{-10}$ and maximum of 200 iterations starting from the $\boldsymbol{\alpha}(\boldsymbol{\beta})$ from the previous iteration. In an outer loop, we monitor the relative decrease in the objective function and terminate when the relative improvement falls below 0.0001. We use a maximum of 40 outer iterations where each iteration performs one round of $\boldsymbol{\alpha}, \boldsymbol{\beta}, \mathbf{Q}$ optimization. Empirically, we find that the block coordinate descent approach often converges surprisingly quickly (see Section 4.2). The final classification is given by Eqn. 5.

## 4 Empirical Study

In this section, we present an empirical study aimed at comparing the proposed algorithms with several baselines: (i) Unsupervised co-clustering with spectral bipartite graph partitioning (BIPARTITE) and non-negative matrix factorization (NMF), (ii) supervised performance of standard regularized least squares classification (RLS) that ignores unlabeled data, and (iii) one-sided semi-supervised performance obtained with Laplacian RLS (LAPRLS) which uses a euclidean nearest-neighbor *row similarity* graph. The goal is to observe whether dual supervision particularly along features can help improve classification performance, and whether joint RKHS regularization as formulated in our algorithms (abbreviated MR for the manifold regularization based method of Section 3.1 and MA for the matrix approximation method of Section 3.2) along both rows and columns leads to good quality out-of-sample prediction. In the experiments below, the performance of RLS and LAPRLS is optimized for best performance on the unlabeled set over a grid of hyperparameters. We use Gaussian kernels with width $\sigma_r$ for rows and $\sigma_c$ for columns. These were set to $2^k \sigma_{0r}, 2^k \sigma_{0c}$ respectively where $\sigma_{0r}, \sigma_{0c}$ are $(1/m)$-quantile of pairwise euclidean distances among rows and columns respectively for an $m$ class problem, and $k$ is tuned over $\{-2, -1, 0, 1, 2\}$ to optimize 3-fold cross-validation performance of fully supervised RLS. The values $\gamma_r, \gamma_c, \mu$ are loosely tuned for MA,MR with respect to a single random split of the data into training and validation set; more careful hyperparameter tuning may further improve the results presented below.

We focus on performance in predicting row labels. To enable comparison with the unsupervised co-clustering methods, we use the popularly used F-measure defined on pairs of examples as follows:

$$\text{Precision} \quad = \quad \frac{\text{Number of Pairs Correctly Predicted}}{\text{Number of Pairs Predicted to be In Same Cluster or Class}}$$

$$\text{Recall} \quad = \quad \frac{\text{Number of Pairs Correctly Predicted}}{\text{Number of Pairs in the Same Cluster or Class}}$$

$$\text{F-measure} \quad = \quad (2 * \text{Precision} * \text{Recall})/(\text{Precision} + \text{Recall}) \qquad (14)$$

### 4.1 A Toy Dataset

We generated a toy 2-class dataset with 200 examples per class and 100 features to demonstrate the main observations. The feature vector for a positive example is of the form $[2\mathbf{u} - 0.1 \; 2\mathbf{u} + 0.1]$, and for a negative example is of the form $[2\mathbf{u} + 0.1 \; 2\mathbf{u} - 0.1]$, where $\mathbf{u}$ is a 50-dimensional random vector whose entries are uniformly distributed over the unit interval. It is clear that there is substantial overlap between the two classes. Given a column partitioning $\pi_c$, consider the following transformation: $T(\boldsymbol{x}) = \left( \frac{\sum_{i:\pi_c(i)=1} x_i}{|i:\pi_c(i)=1|}, \frac{\sum_{i:\pi_c(i)=-1} x_i}{|i:\pi_c(i)=-1|} \right)$ that maps examples in $\Re^{100}$ to the plane $\Re^2$ by composing a single feature whose value equals the mean of all features in the same partition. For the correct column partitioning, $\pi_c(i) = 1, 1 \le i \le 50, \pi_c(i) = -1, 50 < i \le 100$, the examples under the

action of $T$ are shown in Figure 1 (left). It is clear that $T$ renders the data to be almost separable. It is therefore natural to attempt to (effectively) learn $T$ in a semi-supervised manner. In Figure 1 (right), we plot the learning curves of various algorithms with respect to increasing number of row and column labels. On this dataset, co-clustering techniques (BIPARTITE, NMF) perform fairly well, and even significantly better than RLS, which has an optimized F-measure of 67% with 25 row labels. With increasing amounts of column labels, the learning curves of MR and MA steadily lift eventually outperforming the unsupervised techniques. The hyperparameters used in this experiment are: $\sigma_r = 2.1, \sigma_c = 4.1, \gamma_r = \gamma_c = 0.001, \mu = 10$ for MR and $0.001$ for MA.

Figure 1: **left**: Examples in the toy dataset under the transformation defined by the correct column partitioning. **right**: Performance comparison – the number of column labels used are marked.

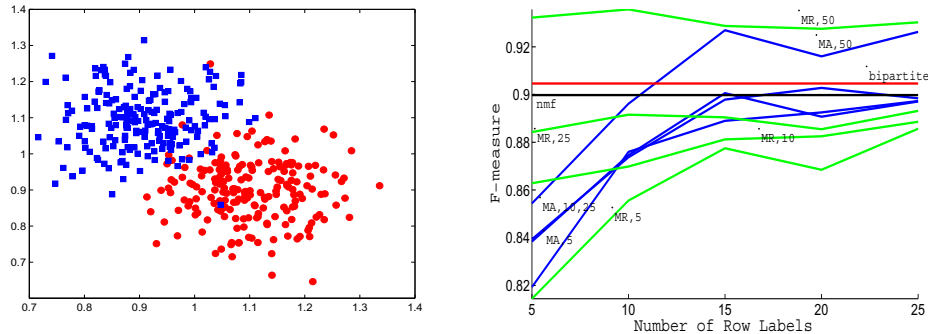

## 4.2    Text Categorization

We performed experiments on document-word matrices drawn from the 20-newsgroups dataset preprocessed as in [15]. The preprocessed data has been made publicly available by the authors of [15][1]. For each word $w$ and class $c$, we computed a score as follows:

score$(w,c) = -P(Y = c) \log P(Y = c) - P(W = w)P(Y = c|W = w) \log P(Y = c|W = w)$ $- P(W \neq w)P(Y = c|W \neq w) \log P(Y = c|W \neq w)$, where $P(Y = c)$ is the fraction of documents whose category is $c$, $P(W = w)$ is the fraction of times word $w$ is encountered, and $P(Y = c|W = w)$ $(P(Y = c|W \neq w))$ is the fraction of documents with class $c$ when $w$ is present (absent). It is easy to see that the mutual information between the indicator random variable for $w$ and the class variable is $\sum_c score(w,c)$. We simulated manual labeling of words by associating $w$ with the class $\text{argmax}_c \, score(w,c)$. Finally, we restricted attention to 631 words with highest overall mutual information and 2000 documents that belong to the following 5 classes: *comp.graphics, rec.motorcycles, rec.sport.baseball, sci.space, talk.politics.mideast*. Since words of *talk.politics.mideast* accounted for more than half the vocabulary, we used the class normalization prescribed in [11] to handle the imbalance in the labeled data.

Results presented in Table 1 are averaged over 10 runs. In each run, we randomly split the documents into training and test sets, in the ratio $1 : 3$. The training set is then further split into labeled and unlabeled sets by randomly selecting 75 labeled documents. We experimented with increasing number of randomly chosen word labels. The hyperparameters are as follows: $\sigma_r = 0.43, \sigma_c = 0.69, \gamma_r = \gamma_c = \mu = 1$ for MR and $\gamma_r = \gamma_c = 0.0001, \mu = 0.01$ for MA.

We observe that even without any word supervision MR outperforms all the baseline approaches: unsupervised co-clustering with BIPARTITE and NMF, standard RLS that only uses labeled documents, and also LAPRLS which uses a graph Laplacian based on document similarity for semi-supervised learning. This validates the effectiveness of the bipartite document and word graph regularizer. As the amount of word supervision increases, the performance of both MR and MA improves gracefully. The out-of-sample extension to test data is of good quality, considering that our test sets are much larger than our training sets. We also observed that the mean number of (outer) iterations required for convergence of MA decreases as labels are increased from 0 to 500: $28.7(0), 12.2(100), 12.7(200), 9.3(350), 7.8(500)$. In, Figure 2 we show the top unlabeled words

Table 1: Performance on 5-Newsgroups Dataset with 75 row labels

(a) F-measure on Unlabeled Set

| BIPARTITE | NMF | RLS | LAPRLS |
|---|---|---|---|
| 54.8 (7.8) | 54.4 (6.2) | 62.2 (3.1) | 62.5 (3.0) |

(b) F-measure on Test Set

| RLS | LAPRLS |
|---|---|
| 61.2 (1.7) | 61.9 (1.4) |

(c) F-measure on Unlabeled Set

| # col labs | MR | MA |
|---|---|---|
| 0 | 64.7 (1.3) | 60.4 (5.6) |
| 100 | 72.3 (2.2) | 59.6 (5.7) |
| 200 | 77.0 (2.5) | 69.2 (7.1) |
| 350 | 78.6 (2.1) | 75.1 (4.1) |
| 500 | 79.3 (1.6) | 77.1 (5.8) |

(d) F-measure on Test Set

| # col labs | MR | MA |
|---|---|---|
| 0 | 57.1 (2.1) | 60.3 (7.0) |
| 100 | 60.9 (2.4) | 60.9 (5.0) |
| 200 | 66.2 (2.8) | 66.2 (6.2) |
| 350 | 68.1 (1.9) | 70.3 (4.4) |
| 500 | 69.1 (2.4) | 71.0 (6.0) |

for each class sorted by the real-valued prediction score assigned by MR (in one run trained with 100 labeled words). Intuitvely, the main words associated with the class are retrieved.

Figure 2: Top unlabeled words categorized by MR

COMP.GRAPHICS: polygon, gifs, conversion, shareware, graphics, rgb, vesa, viewers, gif, format, viewer, amiga, raster, ftp, jpeg, manipulation

REC.MOTORCYCLES: biker, archive, dogs, yamaha, plo, wheel, riders, motorcycle, probes, ama, rockies, neighbors, saudi, kilometers

REC.SPORT.BASEBALL: clemens, morris, pitched, hr, batters, dodgers, offense, reds, rbi, wins, mets, innings, ted, defensive, sox, inning

SCI.SPACE: oo, servicing, solar, scispace, scheduled, atmosphere, missions, telescope, bursts, orbiting, energy, observatory, island, hst, dark

TALK.POLITICS.MIDEAST:turkish, greek, turkey, hezbollah, armenia, territory, ohanus, appressian, sahak, melkonian, civilians, greeks

## 4.3 Project Categorization

We also considered a problem that arises in a real business-intelligence setting. The dataset is composed of 1169 projects tracked by the Integrated Technology Services division of IBM. These projects need to be categorized into 8 predefined product categories within IBM's Server Services product line, with the eventual goal of performing various follow-up business analyses at the granularity of categories. Each project is represented as a 112-dimensional vector specifying the distribution of skills required for its delivery. Therefore, each feature is associated with a particular job role/skill set (JR/SS) combination, e.g., "data-specialist (oracle database)". Domain experts validated project (row) labels and additionally provided category labels for 25 features deemed to be important skills for delivering projects in the corresponding category. By demonstrating our algorithms on this dataset, we are able to validate a general methodology with which to approach project categorization across all service product lines (SPLs) on a regular basis. The amount of dual supervision available in other SPLs is indeed severely limited as both the project categories and skill definitions are constantly evolving due to the highly dynamic business environment.

Results presented in Table 2 are averaged over 10 runs. In each run, we randomly split the projects into training and test sets, in the ratio $3 : 1$. The training set is then further split into labeled and unlabeled sets by randomly selecting 30 labeled projects. We experimented with increasing number of randomly chosen column labels, from none to all 25 available labels. The hyperparameters are as follows: $\gamma_r = \gamma_c = 0.0001, \sigma_r = 0.69, \sigma_c = 0.27$ chosen as described earlier. Results in Tables 2(c),2(d) are obtained with $\mu = 10$ for MR, $\mu = 0.001$ for MA.

We observe that BIPARTITE performs significantly better than NMF on this dataset, and is competitve with supervised RLS performance that relies only on labeled data. By using LAPRLS , performance can be slightly boosted. We find that MR outperforms all approaches significantly even with very few column labels. We conjecture that the comparatively lower mean and high variance in the performance of MA on this dataset is due to suboptimal local minima issues, which may be alleviated using annealing techniques or multiple random starts, commonly used for Transductive SVMs [3]. From Tables 2(c),2(d) we also observe that both methods give high quality out-of-sample extension on this problem.

Table 2: Performance on IBM Project Categorization Dataset with 30 row labels

(a) F-measure on Unlabeled Set

| BIPARTITE | NMF | RLS | LAPRLS |
|---|---|---|---|
| 89.1 (2.7) | 56.5 (1.1) | 88.1 (7.3) | 90.20 (5.8) |

(b) F-measure on Test Set

| RLS | LAPRLS |
|---|---|
| 87.8 (8.4) | 90.2 (6.0) |

(c) F-measure on Unlabeled Set

| # col labs | MR | MA |
|---|---|---|
| 0 | 92.7 (4.6) | 90.7 (4.8) |
| 5 | 94.9 (1.8) | 87.8 (6.4) |
| 10 | 93.0 (4.2) | 89.0 (8.0) |
| 15 | 92.3 (7.0) | 89.1 (7.4) |
| 25 | 98.0 (0.5) | 92.2 (6.0) |

(d) F-measure on Test Set

| # col labs | MR | MA |
|---|---|---|
| 0 | 89.2 (5.5) | 90.0 (5.5) |
| 5 | 93.3 (1.7) | 87.4 (6.6) |
| 10 | 91.9 (4.2) | 89.1 (8.3) |
| 15 | 92.2 (5.2) | 89.2 (8.8) |
| 25 | 96.4 (1.6) | 92.1 (6.8) |

## 5 Conclusion

We have developed semi-supervised kernel methods that support partial supervision along both dimensions of the data. Empirical studies show promising results and highlight the previously untapped benefits of feature supervision in semi-supervised settings. For an application of closely related algorithms to blog sentiment classification, we point the reader to [14]. For recent work on text categorization with labeled features instead of labeled examples, see [8].

## Footnotes

[1]At http://www.princeton.edu/~nslonim/data/20NG_data_74000.mat.gz

## References

[1] A. Banerjee, I. Dhillon, J. Ghosh, S.Merugu, and D.S. Modha. A generalized maximum entropy approach to bregman co-clustering and matrix approximation. *JMLR*, 8:1919–1986, 2007.

[2] M. Belkin, P. Niyogi, and V. Sindhwani. Manifold regularization: A geometric framework for learning from labeled and unlabeled examples. *JMLR*, 7:2399–2434, 2006.

[3] O. Chapelle, B. Schölkopf, and A. Zien, editors. *Semi-Supervised Learning*. MIT Press, 2006.

[4] F. Chung, editor. *Spectral Graph Theory*. AMS, 1997.

[5] I. Dhillon. Co-clustering documents and words using bipartite spectral graph partitioning. In *KDD*, 2001.

[6] C. Ding, X. He, and H.D. Simon. On the equivalence of nonnegative matrix factorization and spectral clustering. In *SDM*, 2005.

[7] C. Ding, T. Li, W. Peng, and H. Park. Orthogonal nonnegative matrix tri-factorizations for clustering. In *KDD*, 2006.

[8] G. Druck, G. Mann, and A. McCallum. Learning from labeled features using generalized expectation criteria. In *SIGIR*, 2008.

[9] J. Gardiner, Laub A.J, Amato J.J, and Moler C.B. Solution of the Sylvester matrix equation $AXB^T + CXD^T = E$. *ACM Transactions on Mathematical Software*, 18(2):223–231, 1992.

[10] D. Harville. *Matrix Algebra From a Statistician's Perspective*. Springer, New York, 1997.

[11] T.M. Huang and V. Kecman. Semi-supervised learning from unbalanced labeled data an improvement. *Lecture Notes in Computer Science*, 3215:765–771, 2004.

[12] A. Langville, C. Meyer, and R. Albright. Initializations for the non-negative matrix factorization. In *KDD*, 2006.

[13] T. Li and C. Ding. The relationships among various nonnegative matrix factorization methods for clustering. In *ICDM*, 2006.

[14] V. Sindhwani and P. Melville. Document-word co-regularization for semi-supervised sentiment analysis. In *ICDM*, 2008.

[15] N. Slonim and N. Tishby. Document clustering using word clusters via the information bottleneck method. In *SIGIR*, 2000.

